# Improving the Performance of Radial Basis Function Networks by Learning Center Locations

**Dietrich Wettschereck**
Department of Computer Science
Oregon State University
Corvallis, OR 97331-3202

**Thomas Dietterich**
Department of Computer Science
Oregon State University
Corvallis, OR 97331-3202

## Abstract

Three methods for improving the performance of (gaussian) radial basis function (RBF) networks were tested on the NETtalk task. In RBF, a new example is classified by computing its Euclidean distance to a set of *centers* chosen by unsupervised methods. The application of supervised learning to learn a non-Euclidean distance metric was found to reduce the error rate of RBF networks, while supervised learning of each center's variance resulted in inferior performance. The best improvement in accuracy was achieved by networks called generalized radial basis function (GRBF) networks. In GRBF, the center locations are determined by supervised learning. After training on 1000 words, RBF classifies 56.5% of letters correct, while GRBF scores 73.4% letters correct (on a separate test set). From these and other experiments, we conclude that supervised learning of center locations can be very important for radial basis function learning.

## 1 Introduction

Radial basis function (RBF) networks are 3-layer feed-forward networks in which each hidden unit $\alpha$ computes the function

$$f_\alpha(\mathbf{x}) = e^{-\frac{||\mathbf{X}-\mathbf{X}_\alpha||^2}{\sigma^2}},$$

and the output units compute a weighted sum of these hidden-unit activations:

$$f^*(\mathbf{x}) = \sum_{\alpha=1}^{N} c_\alpha f_\alpha(\mathbf{x}).$$

In other words, the value of $f^*(\mathbf{x})$ is determined by computing the Euclidean distance between $\mathbf{x}$ and a set of $N$ **centers**, $\mathbf{x}_\alpha$. These distances are then passed through Gaussians (with variance $\sigma^2$ and zero mean), weighted by $c_\alpha$, and summed.

Radial basis function networks (RBF networks) provide an attractive alternative to sigmoid networks for learning real-valued mappings: (a) they provide excellent approximations to smooth functions (Poggio & Girosi, 1989), (b) their "centers" are interpretable as "prototypes", and (c) they can be learned very quickly, because the center locations ($\mathbf{x}_\alpha$) can be determined by unsupervised learning algorithms and the weights ($c_\alpha$) can be computed by pseudo-inverse methods (Moody and Darken, 1989).

Although the application of unsupervised methods to learn the center locations does yield very efficient training, there is some evidence that the generalization performance of RBF networks is inferior to sigmoid networks. Moody and Darken (1989), for example, report that their RBF network must receive 10 times more training data than a standard sigmoidal network in order to attain comparable generalization performance on the Mackey-Glass time-series task.

There are several plausible explanations for this performance gap. First, in sigmoid networks, all parameters are determined by supervised learning, whereas in RBF networks, typically only the learning of the output weights has been supervised. Second, the use of Euclidean distance to compute $\|\mathbf{x} - \mathbf{x}_\alpha\|$ assumes that all input features are equally important. In many applications, this assumption is known to be false, so this could yield poor results.

The purpose of this paper is twofold. First, we carefully tested the performance of RBF networks on the well-known NETtalk task (Sejnowski & Rosenberg, 1987) and compared it to the performance of a wide variety of algorithms that we have previously tested on this task (Dietterich, Hild, & Bakiri, 1990). The results confirm that there is a substantial gap between RBF generalization and other methods.

Second, we evaluated the benefits of employing supervised learning to learn (a) the center locations $\mathbf{x}_\alpha$, (b) weights $w_i$ for a weighted distance metric, and (c) variances $\sigma_\alpha^2$ for each center. The results show that supervised learning of the center locations and weights improves performance, while supervised learning of the variances or of combinations of center locations, variances, and weights did not. The best performance was obtained by supervised learning of only the center locations (and the output weights, of course).

In the remainder of the paper we first describe our testing methodology and review the NETtalk domain. Then, we present results of our comparison of RBF with other methods. Finally, we describe the performance obtained from supervised learning of weights, variances, and center locations.

## 2   Methodology

All of the learning algorithms described in this paper have several parameters (such as the number of centers and the criterion for stopping training) that must be specified by the user. To set these parameters in a principled fashion, we employed the cross-validation methodology described by Lang, Hinton & Waibel (1990). First, as

usual, we randomly partitioned our dataset into a training set and a test set. Then, we further divided the training set into a subtraining set and a cross-validation set. Alternative values for the user-specified parameters were then tried while training on the subtraining set and testing on the cross-validation set. The best-performing parameter values were then employed to train a network on the full training set. The generalization performance of the resulting network is then measured on the test set. Using this methodology, no information from the test set is used to determine any parameters during training.

We explored the following parameters: (a) the number of hidden units (centers) $N$, (b) the method for choosing the initial locations of the centers, (c) the variance $\sigma^2$ (when it was not subject to supervised learning), and (d) (whenever supervised training was involved) the stopping squared error per example. We tried $N = 50, 100, 150, 200$, and $250$; $\sigma^2 = 1, 2, 4, 5, 10, 20$, and $50$; and three different initialization procedures:

(a) Use a subset of the training examples,

(b) Use an unsupervised version of the IB2 algorithm of Aha, Kibler & Albert (1991), and

(c) Apply k-means clustering, starting with the centers from (a).

For all methods, we applied the pseudo-inverse technique of Penrose (1955) followed by Gaussian elimination to set the output weights.

To perform supervised learning of center locations, feature weights, and variances, we applied conjugate-gradient optimization. We modified the conjugate-gradient implementation of backpropagation supplied by Barnard & Cole (1989).

## 3    The NETtalk Domain

We tested all networks on the NETtalk task (Sejnowski & Rosenberg, 1987), in which the goal is to learn to pronounce English words by studying a dictionary of correct pronunciations. We replicated the formulation of Sejnowski & Rosenberg in which the task is to learn to map each individual letter in a word to a phoneme and a stress.

Two disjoint sets of 1000 words were drawn at random from the NETtalk dictionary of 20,002 words (made available by Sejnowski and Rosenberg): one for training and one for testing. The training set was further subdivided into an 800-word subtraining set and a 200-word cross-validation set.

To encode the words in the dictionary, we replicated the encoding of Sejnowski & Rosenberg (1987): Each input vector encodes a 7-letter window centered on the letter to be pronounced. Letters beyond the ends of the word are encoded as blanks. Each letter is locally encoded as a 29-bit string (26 bits for each letter, 1 bit for comma, space, and period) with exactly one bit on. This gives 203 input bits, seven of which are 1 while all others are 0.

Each phoneme and stress pair was encoded using the 26-bit distributed code developed by Sejnowski & Rosenberg in which the bit positions correspond to distinctive features of the phonemes and stresses (e.g., voiced/unvoiced, stop, etc.).

## 4    RBF Performance on the NETtalk Task

We began by testing RBF on the NETtalk task. Cross-validation training determined that peak RBF generalization was obtained with $N = 250$ (the number of centers), $\sigma^2 = 5$ (constant for all centers), and the locations of the centers computed by k-means clustering. Table 1 shows the performance of RBF on the 1000-word test set in comparison with several other algorithms: nearest neighbor, the decision tree algorithm ID3 (Quinlan, 1986), sigmoid networks trained via backpropagation (160 hidden units, cross-validation training, learning rate 0.25, momentum 0.9), Wolpert's (1990) HERBIE algorithm (with weights set via mutual information), and ID3 with error-correcting output codes (ECC, Dietterich & Bakiri, 1991).

Table 1: Generalization performance on the NETtalk task.

| Algorithm | % correct (1000-word test set) | | | |
|---|---|---|---|---|
| | Word | Letter | Phoneme | Stress |
| Nearest neighbor | 3.3 | 53.1 | 61.1 | 74.0 |
| RBF | 3.7 | 57.0***** | 65.6***** | 80.3***** |
| ID3 | 9.6***** | 65.6***** | 78.7***** | 77.2***** |
| Back propagation | 13.6** | 70.6***** | 80.8**** | 81.3***** |
| Wolpert | 15.0 | 72.2* | 82.6***** | 80.2 |
| ID3 + 127-bit ECC | 20.0*** | 73.7* | 85.6***** | 81.1 |

Prior row different, $p < .05^*$  $.01^{**}$  $.005^{***}$  $.002^{****}$  $.001^{*****}$

Performance is shown at several levels of aggregation. The "stress" column indicates the percentage of stress assignments correctly classified. The "phoneme" column shows the percentage of phonemes correctly assigned. A "letter" is correct if the phoneme and stress are correctly assigned, and a "word" is correct if all letters in the word are correctly classified. Also shown are the results of a two-tailed test for the difference of two proportions, which was conducted for each row and the row preceding it in the table.

From this table, it is clear that RBF is performing substantially below virtually all of the algorithms except nearest neighbor. There is certainly room for supervised learning of RBF parameters to improve on this.

## 5    Supervised Learning of Additional RBF Parameters

In this section, we present our supervised learning experiments. In each case, we report only the cross-validation performance. Finally, we take the best supervised learning configuration, as determined by these cross-validation scores, train it on the entire training set and evaluate it on the test set.

### 5.1    Weighted Feature Norm and Centers With Adjustable Widths

The first form of supervised learning that we tested was the learning of a weighted norm. In the NETtalk domain, it is obvious that the various input features are not equally important. In particular, the features describing the letter at the center of

the 7-letter window—the letter to be pronounced—are much more important than the features describing the other letters, which are only present to provide context. One way to capture the importance of different features is through a weighted norm:

$$||x - x_\alpha||_w^2 = \sum_i w_i (x_i - x_{\alpha i})^2.$$

We employed supervised training to obtain the weights $w_i$. We call this configuration $RBF_{FW}$. On the cross-validation set, $RBF_{FW}$ correctly classified 62.4% of the letters ($N=200$, $\sigma^2 = 5$, center locations determined by k-means clustering). This is a 4.7 percentage point improvement over standard RBF, which on the cross-validation set classifies only 57.7% of the letters correctly ($N=250$, $\sigma^2 = 5$, center locations determined by k-means clustering).

Moody & Darken (1989) suggested heuristics to set the variance of each center. They employed the inverse of the mean Euclidean distance from each center to its P-nearest neighbors to determine the variance. However, they found that in most cases a global value for all variances worked best. We replicated this experiment for P = 1 and P = 4, and we compared this to just setting the variances to a global value ($\sigma^2 = 5$) optimized by cross-validation. The performance on the cross-validation set was 53.6% (for P=1), 53.8% (for P=4), and 57.7% (for the global value).

In addition to these heuristic methods, we also tried supervised learning of the variances alone (which we call $RBF_\sigma$). On the cross-validation set, it classifies 57.4% of the letters correctly, as compared with 57.7% for standard RBF.

Hence, in all of our experiments, a single global value for $\sigma^2$ gives better results than any of the techniques for setting separate values for each center. Other researchers have obtained experimental results in other domains showing the usefulness of nonuniform variances. Hence, we must conclude that, while $RBF_\sigma$ did not perform well in the NETtalk domain, it may be valuable in other domains.

## 5.2  Learning Center Locations (Generalized Radial Basis Functions)

Poggio and Girosi (1989) suggest using gradient descent methods to implement supervised learning of the center locations, a method that they call generalized radial basis functions (GRBF). We implemented and tested this approach. On the cross-validation set, GRBF correctly classifies 72.2% of the letters ($N = 200$, $\sigma^2 = 4$, centers initialized to a subset of training data) as compared to 57.7% for standard RBF. This is a remarkable 14.5 percentage-point improvement.

We also tested GRBF with previously learned feature weights ($GRBF_{FW}$) and in combination with learning variances ($GRBF_\sigma$). The performance of both of these methods was inferior to GRBF. For $GRBF_{FW}$, gradient search on the center locations failed to significantly improved performance of $RBF_{FW}$ networks ($RBF_{FW}$ 62.4% vs. $GRBF_{FW}$ 62.8%, $RBF_{FW}$ 54.5% vs. $GRBF_{FW}$ 57.9%). This shows that through the use of a non-Euclidian, fixed metric found by $RBF_{FW}$ the gradient search of $GRBF_{FW}$ is getting caught in a local minimum. One explanation for this is that feature weights and adjustable centers are two alternative ways of achieving the same effect—namely, of making some features more important than others. Redundancy can easily create local minima. To understand this explanation, consider the plots in Figure 1. Figure 1(A) shows the weights of the input features as they

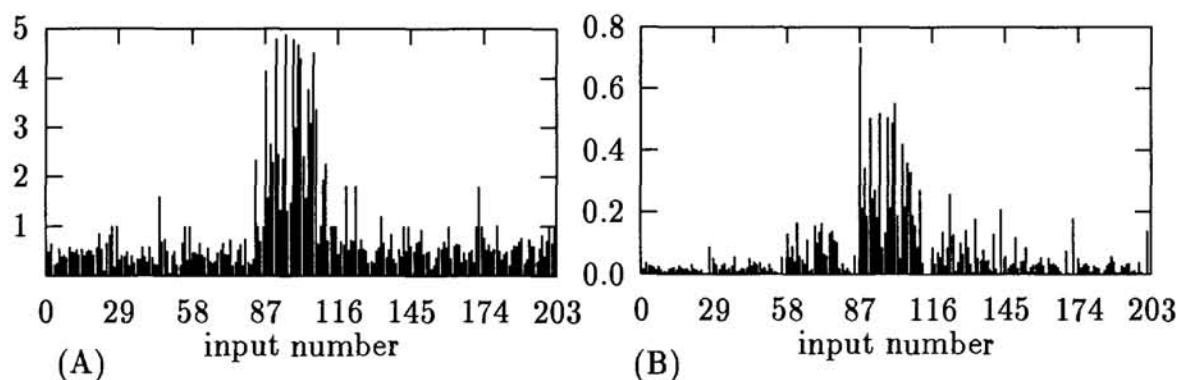

Figure 1: (A) displays the weights of input features as learned by RBF$_{FW}$. In (B) the mean square-distance between centers (separate for each dimension) from a *GRBF* network ($N = 100$, $\sigma^2 = 4$) is shown.

were learned by RBF$_{FW}$. Features with weights near zero have no influence in the distance calculation when a new test example is classified. Figure 1(B) shows the mean squared distance between every center and every other center (computed separately for each input feature). Low values for the mean squared distance on feature $i$ indicate that most centers have very similar values on feature $i$. Hence, this feature can play no role in determining which centers are activated by a new test example. In both plots, the features at the center of the window are clearly the most important. Therefore, it appears that GRBF is able to capture the information about the relative importance of features without the need for feature weights.

To explore the effect of learning the variances and center locations simultaneously, we introduced a scale factor to allow us to adjust the relative magnitudes of the gradients. We then varied this scale factor under cross validation. Generally, the larger we set the scale factor (to increase the gradient of the variance terms) the worse the performance became. As with GRBF$_{FW}$, we see that difficulties in gradient descent training are preventing us from finding a global minimum (or even re-discovering known local minima).

## 5.3   Summary

Based on the results of this section as summarized in Table 2, we chose GRBF as the best supervised learning configuration and applied it to the entire 1000-word training set (with testing on the 1000-word test set). We also combined it with a 63-bit error-correcting output code to see if this would improve its performance, since error-correcting output codes have been shown to boost the performance of backpropagation and ID3. The final comparison results are shown in Table 3. The results show that GRBF is superior to RBF at all levels of aggregation. Furthermore, GRBF is statistically indistinguishable from the best method that we have tested to date (ID3 with 127-bit error-correcting output code), except on phonemes where it is detectably inferior and on stresses where it is detectably superior. GRBF with error-correcting output codes is statistically indistinguishable from ID3 with error-correcting output codes.

Table 2: Percent of letters correctly classified on the 200-word cross-validation data set.

| Method | % Letters Correct |
|--------|--------|
| RBF | 57.7 |
| RBF$_{FW}$ | 62.4 |
| RBF$_\sigma$ | 57.4 |
| GRBF | 72.2 |
| GRBF$_{FW}$ | 62.8 |
| GRBF$_\sigma$ | 67.5 |

Table 3: Generalization performance on the NETtalk task.

| Algorithm | % correct (1000-word test set) | | | |
|-----------|------|--------|---------|--------|
| | Word | Letter | Phoneme | Stress |
| RBF | 3.7 | 57.0 | 65.6 | 80.3 |
| GRBF | 19.8*** | 73.8*** | 84.1*** | 82.4** |
| ID3 + 127-bit ECC | 20.0 | 73.7 | 85.6* | 81.1* |
| GRBF + 63-bit ECC | 19.2 | 74.6 | 85.3 | 82.2 |

Prior row different, $p < .05^*$  $.002^{**}$  $.001^{***}$

The near-identical performance of GRBF and the error-correcting code method and the fact that the use of error correcting output codes does not improve GRBF's performance significantly, suggests that the "bias" of GRBF (i.e., its implicit assumptions about the unknown function being learned) is particularly appropriate for the NETtalk task. This conjecture follows from the observation that error-correcting output codes provide a way of recovering from improper bias (such as the bias of ID3 in this task). This is somewhat surprising, since the mathematical justification for GRBF is based on the smoothness of the unknown function, which is certainly violated in classification tasks.

## 6   Conclusions

Radial basis function networks have many properties that make them attractive in comparison to networks of sigmoid units. However, our tests of RBF learning (unsupervised learning of center locations, supervised learning of output-layer weights) in the NETtalk domain found that RBF networks did not generalize nearly as well as sigmoid networks. This is consistent with results reported in other domains.

However, by employing supervised learning of the center locations as well as the output weights, the GRBF method is able to substantially exceed the generalization performance of sigmoid networks. Indeed, GRBF matches the performance of the best known method for the NETtalk task: ID3 with error-correcting output codes, which, however, is approximately 50 times faster to train.

We found that supervised learning of feature weights (alone) could also improve the performance of RBF networks, although not nearly as much as learning the center locations. Surprisingly, we found that supervised learning of the variances of the Gaussians located at each center hurt generalization performance. Also, combined supervised learning of center locations and feature weights did not perform as well as supervised learning of center locations alone. The training process is becoming stuck in local minima. For GRBF$_{FW}$, we presented data suggesting that feature weights are redundant and that they could be introducing local minima as a result.

Our implementation of GRBF, while efficient, still gives training times comparable to those required for backpropagation training of sigmoid networks. Hence, an

important open problem is to develop more efficient methods for supervised learning of center locations.

While the results in this paper apply only to the NETtalk domain, the markedly superior performance of GRBF over RBF suggests that in new applications of RBF networks, it is important to consider supervised learning of center locations in order to obtain the best generalization performance.

## Acknowledgments

This research was supported by a grant from the National Science Foundation Grant Number IRI-86-57316.

## References

D. W. Aha, D. Kibler & M. K. Albert. (1991) Instance-based learning algorithms. *Machine Learning* 6(1):37-66.

E. Barnard & R. A. Cole. (1989) A neural-net training program based on conjugate-gradient optimization. Rep. No. CSE 89-014. Oregon Graduate Institute, Beaverton, OR.

T. G. Dietterich & G. Bakiri. (1991) Error-correcting output codes: A general method for improving multiclass inductive learning programs. *Proceedings of the Ninth National Conference on Artificial Intelligence (AAAI-91)*, Anaheim, CA: AAAI Press.

T. G. Dietterich, H. Hild, & G. Bakiri. (1990) A comparative study of ID3 and back-propagation for English text-to-speech mapping. *Proceedings of the 1990 Machine Learning Conference*, Austin, TX. 24–31.

K. J. Lang, A. H. Waibel & G. E. Hinton. (1990) A time-delay neural network architecture for isolated word recognition. *Neural Networks* 3:33-43.

J. MacQueen. (1967) Some methods of classification and analysis of multivariate observations. In LeCam, L. M. & Neyman, J. (Eds.), *Proceedings of the 5th Berkeley Symposium on Mathematics, Statistics, and Probability* (p. 281). Berkeley, CA: University of California Press.

J. Moody & C. J. Darken. (1989) Fast learning in networks of locally-tuned processing units. *Neural Computation* 1(2):281-294.

R. Penrose. (1955) A generalized inverse for matrices. *Proceedings of Cambridge Philosophical Society* 51:406-413.

T. Poggio & F. Girosi. (1989) A theory of networks for approximation and learning. Report Number AI-1140. MIT Artificial Intelligence Laboratory, Cambridge, MA.

J. R. Quinlan. (1986) Induction of decision trees. *Machine Learning* 1(1):81-106.

T. J. Sejnowski & C. R. Rosenberg. (1987) Parallel networks that learn to pronounce English text. *Complex Systems* 1:145-168.

D. Wolpert. (1990) Constructing a generalizer superior to NETtalk via a mathematical theory of generalization. *Neural Networks* 3:445-452.